# Location Estimation with a Differential Update Network

**Ali Rahimi and Trevor Darrell**
Artificial Intelligence Laboratory
Massachusetts Institute of Technology
Cambridge, MA 02139
{ali,trevor}@mit.edu

## Abstract

Given a set of hidden variables with an a-priori Markov structure, we derive an online algorithm which approximately updates the posterior as pairwise measurements between the hidden variables become available. The update is performed using Assumed Density Filtering: to incorporate each pairwise measurement, we compute the optimal Markov structure which represents the true posterior and use it as a prior for incorporating the next measurement. We demonstrate the resulting algorithm by calculating globally consistent trajectories of a robot as it navigates along a 2D trajectory. To update a trajectory of length $t$, the update takes $O(t)$. When all conditional distributions are linear-Gaussian, the algorithm can be thought of as a Kalman Filter which simplifies the state covariance matrix after incorporating each measurement.

## 1   Introduction

Consider a hidden Markov chain. Given a sequence of pairwise measurements between the elements of the chain (for example, their differences, corrupted by noise) we are asked to refine our estimate of their values online, as these pairwise measurements become available. We propose the Differential Update Network as a mechanism for solving this problem. We use this mechanism to recover the trajectory of a robot given noisy measurements of its movement between points in its trajecotry. These pairwise displacements are thought of as noise corrupted measurements between the true but unknown poses to be recovered. The recovered trajectories are consistent in the sense that when the camera returns to an already visited position, its estimated pose is consistent with the pose recovered on the earlier visit.

Pose change measurements between two points on the trajectory are obtained by bringing images of the environment acquired at each pose into registration with each other. The required transformation to affect the registration is the pose change measurement. There is a rich literature on computing pose changes from a pair of scans from an optical sensor: 2D [5, 6] and 3D transformations [7, 8, 9] from monocular cameras, or 3D transformations from range imagery [10, 11, 12] are a few examples. These have been used by [1, 2] in 3D model acquisition and by [3, 4] in robot navigation. The trajectory of the robot is defined as the unknown pose from which each frame was acquired, and is maintained in a state vector which is updated as pose changes are measured.

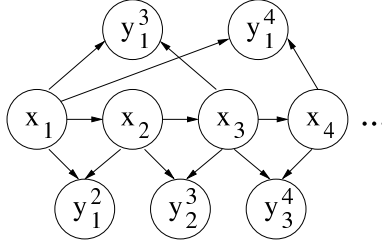

Figure 1: Independence structure of a differential update network.

An alternative method estimates the pose of the robot with respect to fixed features in the world. These methods represent the world as a set of features, such as corners, lines, and other geometric shapes in 3D [13, 14, 15] and match features between a scan at the current pose and the acquired world representation. However, measurements are still pairwise, since they depend on a feature and the poses of the camera. Because both the feature list and the poses are maintained in the state vector, the differential Update Framework can be applied to both scan-based methods and feature-based methods.

Our algorithm incorporates each pose change measurement by updating the pose associated with every frame encountered. To insure that each update can happen in time linear to the length of the trajectory, the correlation structure of the state vector is approximated with a simpler Markov chain after each measurement. This scheme can be thought of as an instance of Assumed Density Filtering (ADF) [16, 17].

The Differential Update Network presented here assumes a linear Gaussian system, but our derivation is general and can accommodate any distribution. For example, we are currently experimenting with discrete distributions. In addition, we focus on frame-based trajectory estimation due to the ready availability of pose change estimators, and to avoid the complexity of maintaining an explicit feature map.

The following section describes the model in a Bayesian framework. Sections 3 and 4 sketch existing batch and online methods for obtaining globally consistent trajectories. Section 5 derives the update rules for our algorithm, which is then applied to a 2D trajectory estimation in section 6.

## 2   Dynamics and Measurement Models

Figure 1 depicts the network. We assume the hidden variables $x_t$ have a Markov structure with known transition densities:

$$p(X) = \prod_{t=1}^{T} p(x_t|x_{t-1}).$$

Pairwise measurements appear on the chain one by one. Conditioned on the hidden variables, these measurements are assumed to be independent:

$$p(Y|X) = \prod_{(s,t) \in M} p(y_s^t|x_s, x_t),$$

where $M$ is the set of pairs of hidden variables which have been measured.

To apply this network to robot localization, let $X = \{x_t\}_{t=1..T}$ be the trajectory of the robot up to time $T$, with each $x_t$ denoting its pose at time $t$. These poses can be represented using any parametrization of pose, for example as 3D rotations and translation, 2D

translations (which is what we use in section 6, or even non-rigid deformations such as affine. The conditional distribution between adjacent $x$'s is assumed to follow:

$$p(x_{t+1}|x_t) = \mathcal{N}(x_{t+1}|x_t, \Lambda_{x|x}). \tag{1}$$

As the robot moves, the pose change estimator computes the motion $y_s^t$ of the robot from two scans of the environment. Given the true poses, we assume that these measurements are independent of each other even when they share a common scan. We model each $y_s^t$ as being drawn from a Gaussian centered around $x_t - x_s$:

$$p(y_s^t|x_s, x_t) = \mathcal{N}(y_s^t|x_t - x_s, \Lambda_{y|xx}) \tag{2}$$

The online global estimation problem requires us to update $p(X|Y)$ as each $y_s^t$ in $Y$ becomes available. The following section reviews a batch solution for computing $p(X|Y)$ using this model. Section 4 discusses a recursive approach with a similar running time as the batch version. Section 5 presents our approach, which performs these updates much faster by simplifying the output of the recursive solution after incorporating each measurement.

## 3  Batch Linear Gaussian Solution

Equation (1) dictates a Gaussian prior $p(X)$ with mean $m_X$ and covariance $\Lambda_X$. Because the pose dynamics are Markovian, the inverse covariance $\Lambda_X^{-1}$ is tri-diagonal. According to equation (2), the observations are drawn from $y_s^t = A_{s,t}X + \omega_{s,t} = x_t - x_s + \omega_{s,t}$, with $\omega_{s,t}$ white and Gaussian with covariance $\lambda_{s,t}$. Stacking up the $A_{s,t}$ and $\lambda_{s,t}$ into $A$ and $\Lambda_{Y|X}$ respectively we know that the posterior mean of $X|Y$ is [21]:

$$m_{X|Y} = m_X + \Lambda_X A^\top \left(A\Lambda_X A^\top + \Lambda_{Y|X}\right)^{-1} Y \tag{3}$$

$$\Lambda_{X|Y} = \Lambda_X - \Lambda_X A^\top \left(A\Lambda_X A^\top + \Lambda_{Y|X}\right)^{-1} A\Lambda_X, \tag{4}$$

or alternatively,

$$\Lambda_{X|Y}^{-1} = \Lambda_X^{-1} + \Lambda_{Y|X}^{-1} \tag{5}$$

$$m_{X|Y} = \Lambda_{X|Y} \left(\Lambda_X^{-1}m_X + \Lambda_{Y|X}^{-1}Y\right). \tag{6}$$

If there are $M$ measurements and $T$ hidden variables, this computation will take $O(T^2M)$ if performed naively. Note that if $M > T$, as is the case in the robot mapping problem, the alternate equations (5) and (6) can be used to obtain a running time of $O(T^3)$.

## 4  Online Linear Gaussian Solution

Lu and Milios [3] proposed a recursive update for updating the trajectory $X|Y^{old}$ after obtaining a new measurement $y_s^t$. Because each measurement is independent of past measurements given the $X$'s, the update is:

$$p(X|Y^{old}, y_s^t) \overset{Bayes}{\propto} p(y_s^t|X)p(X|Y^{old}). \tag{7}$$

Using equations (3) and (4) to perform this update for one $y_s^t$ takes $O(T^2)$. After integrating $M$ measurements, this yields the same final cost as the batch update.

One way to lower this cost is to reduce the number of hidden variables $x_t$ by fixing some of them, thus reducing $T$ [23]. It is also possible to take advantage of the sparseness of the covariance structure of $X|Y^{old}$ by using the updates (6) and (5):

$$\Lambda_{X|new}^{-1}m_{X|new} = \left(\Lambda_{X|old}^{-1}m_{X|old} + \lambda_{y_s^t|old}y_s^t\right) \tag{8}$$

$$\Lambda_{X|new}^{-1} = \Lambda_{X|old}^{-1} + A_{s,t}^\top \lambda_{X|old}^{-1} A_{s,t} \tag{9}$$

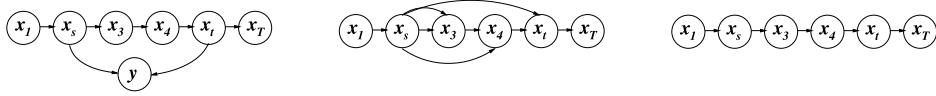

Figure 2: The measurement (left) correlates the hidden variables (middle), whose correlation is then simplified (right), and is ready to accept a new measurement.

Because $\Lambda_{X|new}^{-1}$ has a sparse structure (see equation (9)), $m_{X|new}$ can be found using a sparse linear system solver [23]. Unfortunately, as measurements are incorporated, $\Lambda_{X|new}^{-1}$ becomes denser due to the accumulation of the rank 1 terms in equation (9), rendering this approach less effective.

In the linear Gaussian case, the Differential Update Network addresses this problem by projecting $\Lambda_{X|new}$ on the closest covariance matrix which has a tri-diagonal inverse. Hence, in solving (8), $\Lambda_{X|new}$ is always tri-diagonal, so $m_{X|new}$ is easy to compute.

## 5 Approximate Online Solution

To implement this idea in the general case, we resort to Assumed Density Filtering (ADF) [16]: we approximate $p(X|Y^{old})$ with a simpler distribution $q(X|Y^{old})$. To incorporate a new measurement $y_s^t$, we apply the update

$$p(X|Y^{new}) \overset{Bayes}{\propto} p(y_s^t|x_s, x_t)q(X|Y^{old}). \qquad (10)$$

This new $p(X|Y^{new})$ has a more complicated independence structure than $q(X|Y^{old})$, so incorporating subsequent measurements would require more work and the resulting posterior would be even hairier. So we approximate it again with a $q(X|Y^{new})$ that has a simpler independence structure. Subsequent measurements can again be incorporated easily using this new $q$. Specifically, we force $q$ to always obey Markovian independence. Figure 5 summarizes this process.

The following section discusses how to find a Markovian $q$ so as to minimize the KL divergence between $p$ and $q$. Section 5.2 shows how to incorporate a pairwise measurement on the resulting Markov chain using equation (10).

### 5.1 Simplifying the independence structure

We would like to approximate an arbitrary distribution which factors according to $p(X) = \prod_t p_t(x_t|\text{Pa}[x_t])$, using one which factors into $q(X) = \prod_t q_t(x_t|\text{Qa}[x_t])$. Here, $\text{Pa}[x_t]$ are the parents of node $x_t$ in the graph prescribed by $p(X)$, and $\text{Qa}[x_t][x_t] = x_{t-1}$ are the parents of node $x_t$ as prescribed by $q(X)$.

The objective is to minimize:

$$q^* = \arg\min_q KL\left(\prod p_t \middle\| \prod q_t\right) = \int_x p(X)\ln\frac{p(X)}{\prod_i q_t(x_t|\text{Qa}[x_t])}. \qquad (11)$$

After some manipulation, it can be shown that:

$$q_t^* = p(x_t|\text{Qa}[x_t]). \qquad (12)$$

This says that the best conditional $q_t$ is built up from the corresponding $p_t$ by marginalizing out the conditions that were removed in the graph. This is not an easy operation to perform in general, but the following section shows how to do it in our case.

## 5.2 Computing posterior transitions on a graph with a single loop

This result suggests a simplification to the update of equation (10). Because the ultimate goal is to compute $q(X|Y^{new})$, not $p(X|Y^{new})$, we only need to compute the posterior transitions $p(x_t|x_{t-1}, Y^{new})$. Thus, we circumvent having to first find $p$ then project it onto $q$. We propose computing these transitions in three steps, one for the transitions to the left of $x_s$, another for the loop, and the third for transitions to the right of $x_t$.

### 5.2.1 Finding $p(x_\tau|x_{\tau-1}, y)$ for $\tau = s..t$

For every $s < \tau < t$, notice that
$$p(y, x_{\tau-1}, x_t)p(x_\tau|x_{\tau-1}, x_t) = p(y, x_{\tau-1}, x_\tau, x_t), \qquad (13)$$
because according to figure 5, $p(x_\tau|x_{\tau-1}, x_t) = p(x_\tau|x_{\tau-1}, x_t, y)$. If we could find this joint distribution for all $\tau$, we could find $p(x_\tau|x_{\tau-1}, y)$ by marginalizing out $x_t$ and normalizing. We could also find $p(x_\tau|y)$ by marginalizing out both $x_t$ and $x_{\tau-1}$, then normalizing. Finally, we could compute $p(y, x_\tau, x_t)$ for the next $\tau$ in the iteration.

So there are two missing pieces: The first is $p(y, x_s, x_t)$ for starting the recursion. Computing this term is easy, because $p(y|x_s, x_t)$ is the given measurement model, and $p(x_s, x_t)$ can be obtained easily from the prior by successively applying the total probability theorem.

The second missing piece is $p(x_\tau|x_{\tau-1}, x_t)$. Note that this quantity does not depend on the measurements and could be computed offline if we wanted to. The recursion for calculating it is:
$$p(x_\tau|x_{\tau-1}, x_t) \overset{Bayes}{\propto} p(x_t|x_\tau)p(x_\tau|x_{\tau-1}) \qquad (14)$$
$$p(x_t|x_\tau) = \int dx_{i+1}\, p(x_t|x_{i+1})p(x_{\tau+1}|x_\tau) \qquad (15)$$

The second equation describes a recursion which starts from $t$ and goes down to $s$. It computes the influence of node $\tau$ on node $t$. Equation (14) is coupled to this equation and uses its output. It involves applying Bayes rule to compute a function of 3 variables. Because of the backward nature of (15), $p(x_\tau|x_{\tau-1}, x_t)$ has to be computed using a pass which runs in the opposite direction of the process of (13).

### 5.2.2 Finding $p(x_\tau|x_{\tau-1}, y)$ for $\tau = 1..s$

Starting from $\tau = s - 1$, compute
$$p(y|x_\tau) = \int dx_{\tau+1}\, p(y|x_{\tau+1})p(x_{\tau+1}|x_\tau)$$
$$p(x_\tau|y) \overset{Bayes}{\propto} p(y|x_\tau)p(x_\tau)$$
$$p(x_\tau|x_{\tau-1}, y) \overset{Bayes}{\propto} p(y|x_\tau)p(x_\tau|x_{\tau-1})$$
The recursion first computes the influence of $x_\tau$ on the observation, then computes the marginal and the transition probability.

### 5.2.3 Finding $p(x_\tau|x_{\tau-1}, y)$ for $\tau = t..T$

Starting from $\tau = t$, compute
$$p(x_\tau|y) = \int dx_{\tau-1}\, p(x_\tau|x_{\tau-1}, y)p(x_{\tau-1}|y)$$
$$p(x_\tau|x_{\tau-1}, y) = p(x_\tau|x_{\tau-1})$$
The second identity follows from the independence structure on the right side of observed nodes.

# 6 Results

We manually navigated a camera rig along two trajectories. The camera faced upward and recorded the ceiling. The robot took about 3 minutes to trace each path, producing about 6000 frames of data for each experiment. The trajectory was pre-marked on the floor so we could revisit specific locations (see the rightmost diagrams of figures 6(a,b)). This was done to make the evaluation of the results simpler. The trajectory estimation worked at frame-rate, although it was processed offline to simplify data acquisition.

In these experiments, the pose parameters were $(x, y)$ locations on the floor. All experiments assume the same Brownian motion dynamics. For each new frame, pose changes were computed with respect to at most three base frames. The selection of base frames was based on a measure of appearance between the current frame and all past frames. The pose change estimator was a Lucas-Kanade optical flow tracker [24]. To compute pose displacements, we computed a robust average of the flow vectors using an iterative outlier rejection scheme. We used the number of inlier flow vectors as a crude estimate of the precision of $p(y_s^t | x_s, x_t)$.

Figures 6(a,b) compare the algorithm presented in this paper against two others. The middle plots compare our algorithm (blue) against the batch algorithm which uses equations (5) and (6) (black). Although our recovered trajectories don't coincide exactly with the batch solutions, like the batch solutions, ours are smooth and consistent.

In contrast, more naive methods of reconstructing trajectories do not exhibit these two desiderata. Estimating the motion of each frame with respect to only the previous base frame yields an unsmooth trajectory (green). Furthermore, loops can't be closed correctly (for example, the robot is not found to return to the origin).

The simplest method of taking into account multiple base frames also fails to meet our requirements. The red trajectory shows what happens when we assume individual poses are independent. This corresponds to using a diagonal matrix to represent the correlation between the poses (instead of the tri-diagonal inverse covariance matrix our algorithm uses). Notice that the resulting trajectory is not smooth, and loops are not well closed.

By taking into account a minimum amount of correlation between frame poses, loops have been closed correctly and the trajectory is correctly found to be smooth.

# 7 Conclusion

We have presented a method for approximately computing the posterior distribution of a set of variables for which only pairwise measurements are available. We call the resulting structure a Differential Update Network and showed how to use Assumed Density Filtering to update the posterior as pairwise measurements become available. The two key insights were 1) how to approximate the posterior at each step to minimize KL divergence, and 2) how to compute transition densities on a graph with a single loop in closed form.

We showed how to estimate globally consistent trajectories for a camera using this framework. In this linear-Gaussian context, our algorithm can be thought of as a Kalman Filter which projects the state information matrix down to a tri-diagonal representation while minimizing the KL divergence between the truth and obtain estimate. Although the example used pose change measurements between scans of the environment, our framework can be applied to feature-based mapping and localization as well.

# References

[1] A. Stoddart and A. Hilton. Registration of multiple point sets. In *IJCV*, pages B40–44, 1996.

**(a)**

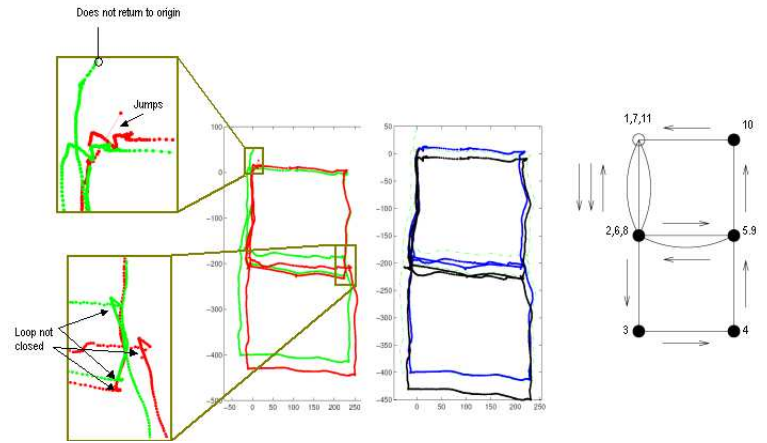

**(b)**

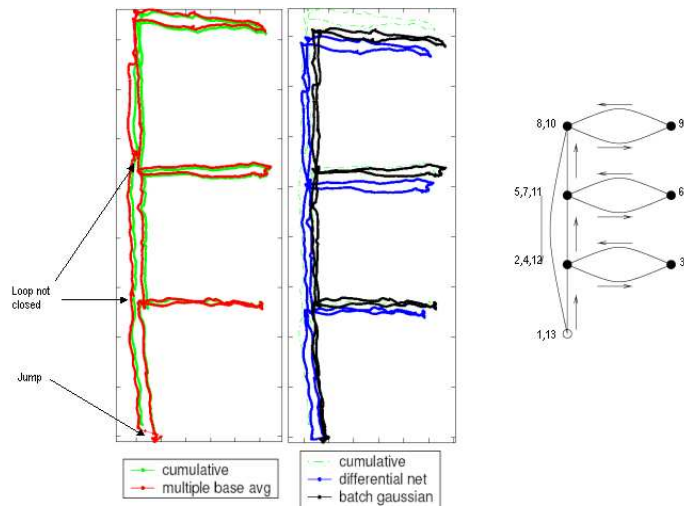

Figure 3: Left, naive accumulation (green) and projecting trajectory to diagonal covariance (red). Loops are not closed well, and trajectory is not smooth. The zoomed areas show that in both naive approaches, there are large jumps in the trajectory, and the pose estimate is incorrect at revisited locations. Right, Differential Update Network (blue) and exact solution (black). Like the batch solution, our solution generates smooth and consistent trajectories.

[2] Y. Chen and G. Medioni. Object modelling by registration of multiple range images. In *Porceedings of the IEEE Internation Conference on Robotics and Authomation*, pages 2724–2728, 1991.

[3] F. Lu and E. Milios. Globally consistent range scan alignment for environment mapping. *Autonomous Robots*, 4:333–349, 1997.

[4] J. Gutmann and K. Konolige. Incremental mapping of large cyclic environments. In *IEEE International Symposium on Computational Intelligence in Robotics and Automation (CIRA)*, 2000.

[5] Harpreet S. Sawhney, Steve Hsu, and Rakesh Kumar. Robust video mosaicing through topology inference and local to global alignment. In *Proc ECCV 2*, pages 103–119, 1998.

[6] H.-Y. Shum and R. Szeliski. Construction of panoramic mosaics with global and local alignment. In *IJCV*, pages 101–130, February 2000.

[7] A. Shashua. Trilinearity in visual recognition by alignment. In *ECCV*, pages 479–484, 1994.

[8] C. Tomasi and T. Kanade. Shape and motion from image streams under orthography: A factorization approach. *International Journal of Computer Vision*, 9(2):137–154, 1992.

[9] Olivier Faugeras. *Three-Dimensional Computer Vision: A Geometric Viewpoint*. MIT Press, Cambridge, Massachusetts, 1993.

[10] M. Harville, A. Rahimi, T. Darrell, G.G. Gordon, and J. Woodfill. 3d pose tracking with linear depth and brightness constraints. In *ICCV99*, pages 206–213, 1999.

[11] Feng Lu and E. Milios. Robot pose estimation in unknown environments by matching 2d range scans. *Robotics and Autonomous Systems*, 22(2):159–178, 1997.

[12] P. J. Besl and N. D. McKay. A method for registration of 3-d shapes. *IEEE Trans. Patt. Anal. Machine Intell.*, 14(2):239–256, February 1992.

[13] N. Ayache and O. Faugeras. Maintaining representations of the environment of a mobile robot. *IEEE Tran. Robot. Automat.*, 5(6):804–819, 1989.

[14] Y. Liu, R. Emery, D. Chakrabarti, W. Burgard, and S. Thrun. Using EM to learn 3D models of indoor environments with mobile robots. In *IEEE International Conference on Machine Learning (ICML)*, 2001.

[15] R. Smith, M. Self, and P. Cheeseman. Estimating uncertain spatial relationships in robotics. In *Uncertainity in Artificial Intelligence*, 1988.

[16] T.P. Minka. Expectation propagation for approximate bayesian inference. In *UAI*, 2001.

[17] X. Boyen and D. Koller. Tractable inference for complex stochastic processes. In *Uncertainty in Artificial Intelligence*, 1998.

[18] T.P. Minka. Independence diagrams. Technical report, Media Lab, http://www.stat.cmu.edu/~minka/papers/diagrams.html, 1998.

[19] J. Pearl. *Probabilistic Reasoning in Intelligent Systems: Networks of Plausible Inference*. Morgan Kaufmann, 1997.

[20] A. Rahimi, L-P. Morency, and T. Darrell. Reducing drift in parametric motion tracking. In *ICCV*, volume 1, pages 315–322, June 2001.

[21] T. Kailath, A. H. Sayed, and B. Hassibi. *Linear Estimation*. Prentice Hall, 2000.

[22] E. Sudderth. Embedded trees: Estimation of gaussian processes on graphs with cycles. Master's thesis, MIT, 2002.

[23] Philip F. McLauchlan. A batch/recursive algorithm for 3d scene reconstruction. *Conf. Computer Vision and Pattern Recognition*, 2:738–743, 2000.

[24] B. D. Lucas and Takeo Kanade. An iterative image registration technique with an application to stereo vision. In *International Joint Conference on Artificial Intelligence*, pages 674–679, 1981.

[25] Andrew W. Fitzgibbon and Andrew Zisserman. Automatic camera recovery for closed or open image sequences. In *ECCV*, pages 311–326, 1998.
